# Bayesian Nonparametric Modeling of Suicide Attempts

**Francisco J. R. Ruiz**
Department of Signal Processing
and Communications
University Carlos III in Madrid
franrruiz@tsc.uc3m.es

**Isabel Valera**
Department of Signal Processing
and Communications
University Carlos III in Madrid
ivalera@tsc.uc3m.es

**Carlos Blanco**
Columbia University College of
Physicians and Surgeons
Cblanco@nyspi.columbia.edu

**Fernando Perez-Cruz**
Department of Signal Processing
and Communications
University Carlos III in Madrid
fernando@tsc.uc3m.es

## Abstract

The National Epidemiologic Survey on Alcohol and Related Conditions (NE-SARC) database contains a large amount of information, regarding the way of life, medical conditions, etc., of a representative sample of the U.S. population. In this paper, we are interested in seeking the hidden causes behind the suicide attempts, for which we propose to model the subjects using a nonparametric latent model based on the Indian Buffet Process (IBP). Due to the nature of the data, we need to adapt the observation model for discrete random variables. We propose a generative model in which the observations are drawn from a multinomial-logit distribution given the IBP matrix. The implementation of an efficient Gibbs sampler is accomplished using the Laplace approximation, which allows integrating out the weighting factors of the multinomial-logit likelihood model. Finally, the experiments over the NESARC database show that our model properly captures some of the hidden causes that model suicide attempts.

## 1 Introduction

Every year, more than 34,000 suicides occur and over 370,000 individuals are treated for self-inflicted injuries in emergency rooms in the U.S., where suicide prevention is one of the top public health priorities [1]. The current strategies for suicide prevention have focused mainly on both the detection and treatment of mental disorders [13], and on the treatment of the suicidal behaviors themselves [4]. However, despite prevention efforts including improvements in the treatment of depression, the lifetime prevalence of suicide attempts in the U.S. has remained unchanged over the past decade [8]. This suggests that there is a need to improve understanding of the risk factors for suicide attempts beyond psychiatric disorders, particularly in non-clinical populations.

According to the National Strategy for Suicide Prevention, an important first step in a public health approach to suicide prevention is to identify those at increased risk for suicide attempts [1]. Suicide attempts are, by far, the best predictor of completed suicide [12] and are also associated with major morbidity themselves [11]. The estimation of suicide attempt risk is a challenging and complex task, with multiple risk factors linked to increased risk. In the absence of reliable tools for identifying those at risk for suicide attempts, be they clinical or laboratory tests, risk detection still relays mainly on clinical variables. The adequacy of the current predictive models and screening methods has been

questioned [12], and it has been suggested that the methods currently used for research on suicide risk factors and prediction models need revamping [9].

Databases that model the behavior of human populations present typically many related questions and analyzing each one of them individually, or a small group of them, do not lead to conclusive results. For example, the National Epidemiologic Survey on Alcohol and Related Conditions (NE-SARC) samples the U.S. population with nearly 3,000 questions regarding, among others, their way of life, their medical conditions, depression and other mental disorders. It contains yes-or-no questions, and some multiple-choice and questions with ordinal answers.

In this paper, we propose to model the subjects in this database using a nonparametric latent model that allows us to seek hidden causes and compact in a few features the immense redundant information. Our starting point is the Indian Buffet Process (IBP) [5], because it allows us to infer which latent features influence the observations and how many features there are. We need to adapt the observation model for discrete random variables, as the discrete nature of the database does not allow us to use the standard Gaussian observation model. There are several options for modeling discrete outputs given the hidden latent features, like a Dirichlet distribution or sampling from the features, but we prefer a generative model in which the observations are drawn from a multinomial-logit distribution because it is similar to the standard Gaussian observation model, where the observation probability distribution depends on the IBP matrix weighted by some factors. Furthermore, the multinomial-logit model, besides its versatility, allows the implementation of an efficient Gibbs sampler where the Laplace approximation [10] is used to integrate out the weighting factors, which can be efficiently computed using the Matrix Inversion Lemma.

The IBP model combined with discrete observations has already been tackled in several related works. In [17], the authors propose a model that combines properties from both the hierarchical Dirichlet process (HDP) and the IBP, called IBP compound Dirichlet (ICD) process. They apply the ICD to focused topic modeling, where the instances are documents and the observations are words from a finite vocabulary, and focus on decoupling the prevalence of a topic in a document and its prevalence in all documents. Despite the discrete nature of the observations under this model, these assumptions are not appropriate for categorical observations such as the set of possible responses to the questions in the NESARC database. Titsias [14] introduced the infinite gamma-Poisson process as a prior probability distribution over non-negative integer valued matrices with a potentially infinite number of columns, and he applies it to topic modeling of images. In this model, each (discrete) component in the observation vector of an instance depends only on one of the active latent features of that object, randomly drawn from a multinomial distribution. Therefore, different components of the observation vector might be equally distributed. Our model is more flexible in the sense that it allows different probability distributions for every component in the observation vector, which is accomplished by weighting differently the latent variables.

## 2 The Indian Buffet Process

In latent feature modeling, each object can be represented by a vector of latent features, and the observations are generated from a distribution determined by those latent feature values. Typically, we have access to the set of observations and the main goal of these models is to find out the latent variables that represent the data. The most common nonparametric tool for latent feature modeling is the Indian Buffet Process (IBP).

The IBP places a prior distribution over binary matrices where the number of columns (features) $K$ is not bounded, i.e., $K \to \infty$. However, given a finite number of data points $N$, it ensures that the number of non-zero columns $K_+$ is finite with probability one. Let $\mathbf{Z}$ be a random $N \times K$ binary matrix distributed following an IBP, i.e., $\mathbf{Z} \sim \mathrm{IBP}(\alpha)$, where $\alpha$ is the concentration parameter of the process. The $n^{th}$ row of $\mathbf{Z}$, denoted by $\mathbf{z}_{n\cdot}$, represents the vector of latent features of the $n^{th}$ data point, and every entry $nk$ is denoted by $z_{nk}$. Note that each element $z_{nk} \in \{0, 1\}$ indicates whether the $k^{th}$ feature contributes to the $n^{th}$ data point.

Given a binary latent feature matrix $\mathbf{Z}$, we assume that the $N \times D$ observation matrix $\mathbf{X}$, where the $n^{th}$ row contains a $D$-dimensional observation vector $\mathbf{x}_{n\cdot}$, is distributed according to a probability distribution $p(\mathbf{X}|\mathbf{Z})$. Additionally, $\mathbf{x}_{\cdot d}$ stands for the $d^{th}$ column of $\mathbf{X}$, and each element of the

matrix is denoted by $x_{nd}$. For instance, in the standard observation model described in [5], $p(\mathbf{X}|\mathbf{Z})$ is a Gaussian probability density function.

MCMC (Markov Chain Monte Carlo) methods have been broadly applied to infer the latent structure $\mathbf{Z}$ from a given observation matrix $\mathbf{X}$ (see, e.g., [5, 17, 15, 14]). In particular, we focus on the use of Gibbs sampling for posterior inference over the latent variables. The algorithm iteratively samples the value of each element $z_{nk}$ given the remaining variables, i.e., it samples from

$$p(z_{nk} = 1|\mathbf{X}, \mathbf{Z}_{\neg nk}) \propto p(\mathbf{X}|\mathbf{Z})p(z_{nk} = 1|\mathbf{Z}_{\neg nk}), \tag{1}$$

where $\mathbf{Z}_{\neg nk}$ denotes all the entries of $\mathbf{Z}$ other than $z_{nk}$. The distribution $p(z_{nk} = 1|\mathbf{Z}_{\neg nk})$ can be readily derived from the exchangeable IBP and can be written as $p(z_{nk} = 1|\mathbf{Z}_{\neg nk}) = m_{-n,k}/N$, where $m_{-n,k}$ is the number of data points with feature $k$, not including $n$, i.e., $m_{-n,k} = \sum_{i \neq n} z_{ik}$.

## 3  Observation model

Let us consider that the observations are discrete, i.e., each element $x_{nd} \in \{1, \ldots, R_d\}$, where this finite set contains the indexes to all the possible values of $x_{nd}$. For simplicity and without loss of generality, we consider that $R_d = R$, but the following results can be readily extended to a different cardinality per input dimension, as well as mixing continuous variables with discrete variables, since given the latent matrix $\mathbf{Z}$ the columns of $\mathbf{X}$ are assumed to be independent.

We introduce matrices $\mathbf{B}^d$ of size $K \times R$ to model the probability distribution over $\mathbf{X}$, such that $\mathbf{B}^d$ links the hidden latent variables with the $d^{th}$ column of the observation matrix $\mathbf{X}$. We assume that the probability of $x_{nd}$ taking value $r$ ($r = 1, \ldots, R$), denoted by $\pi_{nd}^r$, is given by the multiple-logistic function, i.e.,

$$\pi_{nd}^r = p(x_{nd} = r|\mathbf{z}_{n\cdot}, \mathbf{B}^d) = \frac{\exp\left(\mathbf{z}_{n\cdot}\mathbf{b}_{\cdot r}^d\right)}{\displaystyle\sum_{r'=1}^{R} \exp\left(\mathbf{z}_{n\cdot}\mathbf{b}_{\cdot r'}^d\right)}, \tag{2}$$

where $\mathbf{b}_{\cdot r}^d$ denotes the $r^{th}$ column of $\mathbf{B}^d$. Note that the matrices $\mathbf{B}^d$ are used to weight differently the contribution of every latent feature for every component $d$, similarly as in the standard Gaussian observation model in [5]. We assume that the mixing vectors $\mathbf{b}_{\cdot r}^d$ are Gaussian distributed with zero mean and covariance matrix $\mathbf{\Sigma}_b = \sigma_B^2\mathbf{I}$.

The choice of the observation model in Eq. 2, which combines the multiple-logistic function with Gaussian parameters, is based on the fact that it induces dependencies among the probabilities $\pi_{nd}^r$ that cannot be captured with other distributions, such as the Dirichlet distribution [2]. Furthermore, this multinomial-logistic normal distribution has been widely used to define probability distributions over discrete random variables (see, e.g., [16, 2]).

We consider that elements $x_{nd}$ are independent given the latent feature matrix $\mathbf{Z}$ and the $D$ matrices $\mathbf{B}^d$. Then, the likelihood for any matrix $\mathbf{X}$ can be expressed as

$$p(\mathbf{X}|\mathbf{Z}, \mathbf{B}^1, \ldots, \mathbf{B}^D) = \prod_{n=1}^{N}\prod_{d=1}^{D} p(x_{nd}|\mathbf{z}_{n\cdot}, \mathbf{B}^d) = \prod_{n=1}^{N}\prod_{d=1}^{D} \pi_{nd}^{x_{nd}}. \tag{3}$$

### 3.1  Laplace approximation for inference

In Section 2, the (heuristic) Gibbs sampling algorithm for the posterior inference over the latent variables of the IBP has been reviewed and it is detailed in [5]. To sample from Eq. 1, we need to integrate out $\mathbf{B}^d$ in (3), as sequentially sampling from the posterior distribution of $\mathbf{B}^d$ is intractable, for which an approximation is required. We rely on the Laplace approximation to integrate out the parameters $\mathbf{B}^d$ for simplicity and ease of implementation. We first consider the finite form of the proposed model, where $K$ is bounded.

Recall that our model assumes independence among the observations given the hidden latent variables. Then, the posterior $p(\mathbf{B}^1, \ldots, \mathbf{B}^D|\mathbf{X}, \mathbf{Z})$ factorizes as

$$p(\mathbf{B}^1, \ldots, \mathbf{B}^D|\mathbf{X}, \mathbf{Z}) = \prod_{d=1}^{D} p(\mathbf{B}^d|\mathbf{x}_{\cdot d}, \mathbf{Z}) = \prod_{d=1}^{D} \frac{p(\mathbf{x}_{\cdot d}|\mathbf{B}^d, \mathbf{Z})p(\mathbf{B}^d)}{p(\mathbf{x}_{\cdot d}|\mathbf{Z})}. \tag{4}$$

Hence, we only need to deal with each term $p(\mathbf{B}^d|\mathbf{x}_{.d}, \mathbf{Z})$ individually. Although the prior $p(\mathbf{B}^d)$ is Gaussian, due to the non-conjugacy with the likelihood term, the computation of the posterior $p(\mathbf{B}^d|\mathbf{x}_{.d}, \mathbf{Z})$ turns out to be intractable. Following a similar procedure as in Gaussian processes for multiclass classification [16], we approximate the posterior $p(\mathbf{B}^d|\mathbf{x}_{.d}, \mathbf{Z})$ as a Gaussian distribution using Laplace's method. In order to obtain the parameters of the Gaussian distribution, we define $\psi(\mathbf{B}^d)$ as the un-normalized log-posterior of $p(\mathbf{B}^d|\mathbf{x}_{.d}, \mathbf{Z})$, i.e.,

$$\psi(\mathbf{B}^d) = \log p(\mathbf{x}_{.d}|\mathbf{B}^d, \mathbf{Z}) + \log p(\mathbf{B}^d)$$

$$= \text{trace}\left\{\mathbf{M}^{d^\top}\mathbf{B}^d\right\} - \sum_{n=1}^{N} \log\left(\sum_{r'=1}^{R} \exp(\mathbf{z}_{n.}\mathbf{b}_{.r'}^d)\right) - \frac{1}{2\sigma_B^2}\text{trace}\left\{\mathbf{B}^{d^\top}\mathbf{B}^d\right\} - \frac{RK}{2}\log(2\pi\sigma_B^2),$$

(5)

where $(\mathbf{M}^d)_{kr}$ counts the number of data points for which $x_{nd} = r$ and $z_{nk} = 1$, namely, $(\mathbf{M}^d)_{kr} = \sum_{n=1}^{N} \delta(x_{nd} = r)z_{nk}$, where $\delta(\cdot)$ is the Kronecker delta function.

As we prove below, the function $\psi(\mathbf{B}^d)$ is a strictly concave function of $\mathbf{B}^d$ and therefore it has a unique maximum, which is reached at $\mathbf{B}_{\text{MAP}}^d$, denoted by the subscript 'MAP' because it coincides with the mean value of the Gaussian distribution in the Laplace's method (MAP stands for maximum *a posteriori*). We apply Newton's method to compute this maximum.

By defining $(\boldsymbol{\rho}^d)_{kr} = \sum_{n=1}^{N} z_{nk}\pi_{nd}^r$, the gradient of $\psi(\mathbf{B}^d)$ can be derived as

$$\nabla\psi = \mathbf{M}^d - \boldsymbol{\rho}^d - \frac{1}{\sigma_B^2}\mathbf{B}^d.$$

(6)

To compute the Hessian, it is easier to define the gradient $\nabla\psi$ as a vector, instead of a matrix, and hence we stack the columns of $\mathbf{B}^d$ into $\boldsymbol{\beta}^d$, i.e., for avid Matlab users, $\boldsymbol{\beta}^d = \mathbf{B}^d(:)$. The Hessian matrix can now be readily computed taking the derivatives of the gradient, yielding

$$\nabla\nabla\psi = -\frac{1}{\sigma_B^2}\mathbf{I}_{RK} + \nabla\nabla\log p(\mathbf{x}_{.d}|\boldsymbol{\beta}^d, \mathbf{Z})$$

$$= -\frac{1}{\sigma_B^2}\mathbf{I}_{RK} - \sum_{n=1}^{N}\left(\text{diag}(\boldsymbol{\pi}_{nd}) - (\boldsymbol{\pi}_{nd})^\top\boldsymbol{\pi}_{nd}\right) \otimes (\mathbf{z}_{n.}^\top\mathbf{z}_{n.}),$$

(7)

where $\boldsymbol{\pi}_{nd} = \begin{bmatrix} \pi_{nd}^1, & \pi_{nd}^2, & \dots, & \pi_{nd}^R \end{bmatrix}$, and $\text{diag}(\boldsymbol{\pi}_{nd})$ is a diagonal matrix with the values of the vector $\boldsymbol{\pi}_{nd}$ as its diagonal elements. The posterior $p(\boldsymbol{\beta}^d|\mathbf{x}_{.d}, \mathbf{Z})$ can be approximated as

$$p(\boldsymbol{\beta}^d|\mathbf{x}_{.d}, \mathbf{Z}) \approx q(\boldsymbol{\beta}^d|\mathbf{x}_{.d}, \mathbf{Z}) = \mathcal{N}(\boldsymbol{\beta}^d|\boldsymbol{\beta}_{\text{MAP}}^d, (-\nabla\nabla\psi)|_{\boldsymbol{\beta}_{\text{MAP}}^d}),$$

(8)

where $\boldsymbol{\beta}_{\text{MAP}}^d$ contains all the columns of $\mathbf{B}_{\text{MAP}}^d$ stacked into a vector.

Since $p(\mathbf{x}_{.d}|\boldsymbol{\beta}^d, \mathbf{Z})$ is a log-concave function of $\boldsymbol{\beta}^d$ (see [3, p. 87]), $-\nabla\nabla\psi$ is a positive definite matrix, which guarantees that the maximum of $\psi(\boldsymbol{\beta}^d)$ is unique. Once the maximum $\mathbf{B}_{\text{MAP}}^d$ has been determined, the marginal likelihood $p(\mathbf{x}_{.d}|\mathbf{Z})$ can be readily approximated by

$$\log p(\mathbf{x}_{.d}|\mathbf{Z}) \approx \log q(\mathbf{x}_{.d}|\mathbf{Z}) = -\frac{1}{2\sigma_B^2}\text{trace}\left\{(\mathbf{B}_{\text{MAP}}^d)^\top\mathbf{B}_{\text{MAP}}^d\right\}$$

$$- \frac{1}{2}\log\left|\mathbf{I}_{RK} + \sigma_B^2\sum_{n=1}^{N}\left(\text{diag}(\widehat{\boldsymbol{\pi}}_{nd}) - (\widehat{\boldsymbol{\pi}}_{nd})^\top\widehat{\boldsymbol{\pi}}_{nd}\right) \otimes (\mathbf{z}_{n.}^\top\mathbf{z}_{n.})\right| + \log p(\mathbf{x}_{.d}|\mathbf{B}_{\text{MAP}}^d, \mathbf{Z}), \quad (9)$$

where $\widehat{\boldsymbol{\pi}}_{nd}$ is the vector $\boldsymbol{\pi}_{nd}$ evaluated at $\mathbf{B}^d = \mathbf{B}_{\text{MAP}}^d$.

Similarly as in [5], it is straightforward to prove that the limit of Eq. 9 is well-defined if $\mathbf{Z}$ has an un-bounded number of columns, i.e., as $K \to \infty$. The resulting expression for the marginal likelihood $p(\mathbf{x}_{.d}|\mathbf{Z})$ can be readily obtained from Eq. 9 by replacing $K$ by $K_+$, $\mathbf{Z}$ by the submatrix containing only the non-zero columns of $\mathbf{Z}$, and $\mathbf{B}_{\text{MAP}}^d$ by the submatrix containing the $K_+$ corresponding rows. Through the rest of the paper, let us denote with $\mathbf{Z}$ the matrix that contains only the $K_+$ non-zero columns of the full IBP matrix.

## 3.2 Speeding up the matrix inversion

The inverse of the Hessian matrix, as well as its determinant in (9), can be efficiently carried out if we rearrange the inverse of $\nabla\nabla\psi$ as follows

$$(-\nabla\nabla\psi)^{-1} = \left(\mathbf{D} - \sum_{n=1}^{N} \mathbf{v}_n \mathbf{v}_n^\top\right)^{-1}, \qquad (10)$$

where $\mathbf{v}_n = (\boldsymbol{\pi}_{nd})^\top \otimes \mathbf{z}_n^\top$ and $\mathbf{D}$ is a block-diagonal matrix, in which each diagonal submatrix is

$$\mathbf{D}_r = \frac{1}{\sigma_B^2}\mathbf{I}_{K_+} + \mathbf{Z}^\top \operatorname{diag}\left(\boldsymbol{\pi}_{\cdot d}^r\right)\mathbf{Z}, \qquad (11)$$

with $\boldsymbol{\pi}_{\cdot d}^r = \begin{bmatrix} \pi_{1d}^r, \ldots, \pi_{Nd}^r \end{bmatrix}^\top$. Since $\mathbf{v}_n \mathbf{v}_n^\top$ is a rank-one matrix, we can apply the Woodbury identity [18] $N$ times to invert the matrix $-\nabla\nabla\psi$, similar to the RLS (Recursive Least Squares) updates [7]. At each iteration $n = 1, \ldots, N$, we compute

$$(\mathbf{D}^{(n)})^{-1} = \left(\mathbf{D}^{(n-1)} - \mathbf{v}_n \mathbf{v}_n^\top\right)^{-1} = (\mathbf{D}^{(n-1)})^{-1} + \frac{(\mathbf{D}^{(n-1)})^{-1}\mathbf{v}_n \mathbf{v}_n^\top (\mathbf{D}^{(n-1)})^{-1}}{1 - \mathbf{v}_n^\top (\mathbf{D}^{(n-1)})^{-1}\mathbf{v}_n}. \qquad (12)$$

For the first iteration, we define $\mathbf{D}^{(0)}$ as the block-diagonal matrix $\mathbf{D}$, whose inverse matrix involves computing the $R$ matrix inversions of size $K_+ \times K_+$ of the matrices in (11), which can be efficiently solved applying the Matrix Inversion Lemma. After $N$ iterations of (12), it turns out that $(-\nabla\nabla\psi)^{-1} = (\mathbf{D}^{(N)})^{-1}$.

For the determinant in (9), similar recursions can be applied using the Matrix Determinant Lemma [6], which states that $|\mathbf{D} + \mathbf{v}\mathbf{u}^\top| = (1 + \mathbf{v}^\top \mathbf{D}\mathbf{u})|\mathbf{D}|$, and $|\mathbf{D}^{(0)}| = \prod_{r=1}^{R} |\mathbf{D}_r|$.

# 4 Experiments

## 4.1 Inference over synthetic images

We generate a simple example inspired by the experiment in [5, p. 1205] to show that the proposed model works as it should. We define four base black-and-white images that can be present or absent with probability 0.5 independently of each other (Figure 1a), which are combined to create a binary composite image. We also multiply each pixel independently with equiprobable binary noise, hence each white pixel in the composite image can be turned black 50% of the times, while black pixels always remain black. Several examples can be found in Figure 1c. We generate 200 examples to learn the IBP model. The Gibbs sampler has been initialized with $K_+ = 2$, setting each $z_{nk} = 1$ with probability $1/2$, and the hyperparameters have been set to $\alpha = 0.5$ and $\sigma_B^2 = 1$.

After 200 iterations, the Gibbs sampler returns four latent features. Each of the four features recovers one of the base images with a different ordering, which is inconsequential. In Figure 1b, we have plotted the posterior probability for each pixel being white, when only one of the components is active. As expected, the black pixels are known to be black (almost zero probability of being white) and the white pixels have about a $50/50$ chance of being black or white, due to the multiplicative noise. The Gibbs sampler has used as many as nine hidden features, but after iteration 60, the first four features represent the base images and the others just lock on a noise pattern, which eventually fades away.

## 4.2 National Epidemiologic Survey on Alcohol and Related Conditions (NESARC)

The NESARC was designed to determine the magnitude of alcohol use disorders and their associated disabilities. Two waves of interviews have been fielded for this survey (first wave in 2001-2002 and second wave in 2004-2005). For the following experimental results, we only use the data from the first wave, for which 43,093 people were selected to represent the U.S. population 18 years of age and older. Public use data are currently available for this wave of data collection.

Through 2,991 entries, the NESARC collects data on the background of participants, alcohol and other drug consumption and abuse, medicine use, medical treatment, mental disorders, phobias,

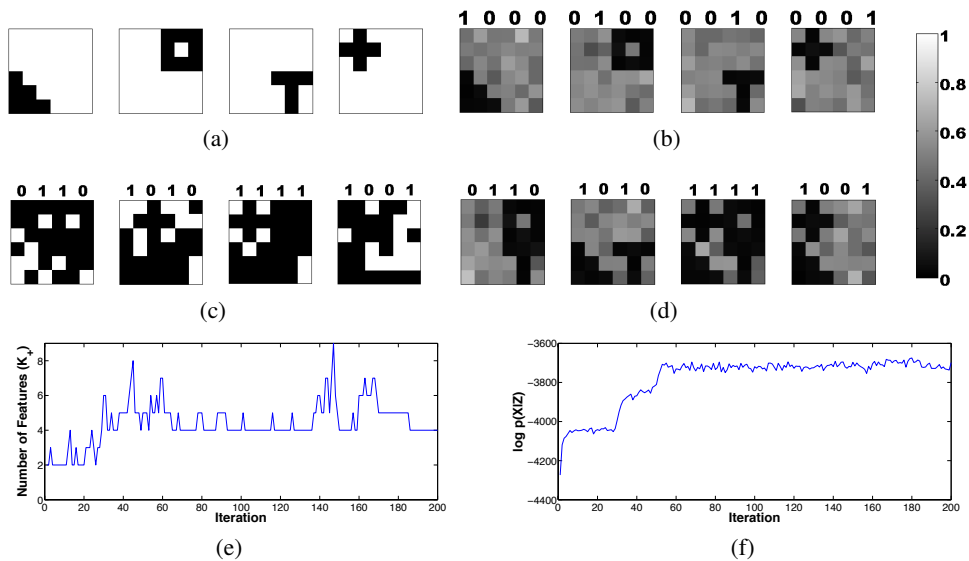

Figure 1: Experimental results of the infinite binary multinomial-logistic model over the image data set. (a) The four base images used to generate the 200 observations. (b) Probability of each pixel being white, when a single feature is active (ordered to match the images on the left), computed using $\mathbf{B}_{\mathrm{MAP}}^d$. (c) Four data points generated as described in the text. The numbers above each figure indicate which features are present in that image. (d) Probabilities of each pixel being white after 200 iterations of the Gibbs sampler inferred for the four data points on (c). The numbers above each figure show the inferred value of $\mathbf{z}_{n\cdot}$ for these data points. (e) The number of latent features $K_+$ and (f) the approximate log of $p(\mathbf{X}|\mathbf{Z})$ over the 200 iterations of the Gibbs sampler.

family history, etc. The survey includes a question about having attempted suicide as well as other related questions such as 'felt like wanted to die' and 'thought a lot about own death'. In the present paper, we use the IBP with discrete observations for a preliminary study in seeking the latent causes which lead to committing suicide. Most of the questions in the survey (over 2,500) are yes-or-no questions, which have four possible outcomes: 'blank' (B), 'unknown' (U), 'yes' (Y) and 'no' (N). If a question is left blank the question was not asked[1]. If a question is said to be unknown either it was not answered or was unknown to the respondent.

In our ongoing study, we want to find a latent model that describes this database and can be used to infer patterns of behavior and, specifically, be able to predict suicide. In this paper, we build an unsupervised model with the 20 variables that present the highest mutual information with the suicide attempt question, which are shown in Table 1 together with their code in the questionnaire.

We run the Gibbs sampler over 500 randomly chosen subjects out of the 13,670 that have answered affirmatively to having had a period of low mood. In this study, we use another 9,500 as test cases and have left the remaining samples for further validation. We have initialized the sampler with an active feature, i.e., $K_+ = 1$, and have set $z_{nk} = 1$ randomly with probability 0.5, and fixing $\alpha = 1$ and $\sigma_B^2 = 1$. After 200 iterations, we obtain seven latent features.

In Figure 2, we have plotted the posterior probability for each question when a single feature is active. In these plots, white means 0 and black 1, and each row sums up to one. Feature 1 is active for modeling the 'blank' and 'no' answers and, fundamentally, those who were not asked Questions 8 and 10. Feature 2 models the 'yes' and 'no' answers and favors affirmative responses to Questions 1, 2, 5, 9, 11, 12, 17 and 18, which indicates depression. Feature 3 models blank answers for most of the questions and negative responses to 1, 2, 5, 8 and 10, which are questions related to suicide. Feature 4 models the affirmative answers to 1, 2, 5, 9 and 11 and also have higher probability for unknowns in Questions 3, 4, 6 and 7. Feature 5 models the 'yes' answer to Questions 3, 4, 6, 7, 8,

10, 17 and 18, being ambivalent in Questions 1 and 2. Feature 6 favors 'blank' and 'no' answers in most questions. Feature 7 models answering affirmatively to Questions 15, 16, 19 and 20, which are related to alcohol abuse.

We show the percentage of respondents that answered positively to the suicide attempt questions in Table 2, independently for the 500 samples that were used to learn the IBP and the 9,500 hold-out samples, together with the total number of respondents. A dash indicates that the feature can be active or inactive. Table 2 is divided in three parts. The first part deals with each individual feature and the other two study some cases of interest. Throughout the database, the prevalence of suicide attempt is 7.83%. As expected, Features 2, 4, 5 and 7 favor suicide attempt risk, although Feature 5 only mildly, and Features 1, 3 and 6 decrease the probability of attempting suicide. From the above description of each feature, it is clear that having Features 4 or 7 active should increase the risk of attempting suicide, while having Features 3 and 1 active should cause the opposite effect.

Features 3 and 4 present the lowest and the highest risk of suicide, respectively, and they are studied together in the second part of Table 2, in which we can see that having Feature 3 and not having Feature 4 reduces this risk by an order of magnitude, and that combination is present in 70% of the population. The other combinations favor an increased rate of suicide attempts that goes from doubling ('11') to quadrupling ('00'), to a ten-fold increase ('01'), and the percentages of population with these features are, respectively, 21%, 6% and 3%.

In the final part of Table 2, we show combinations of features that significantly increase the suicide attempt rate for a reduced percentage of the population, as well as combinations of features that significantly decrease the suicide attempt rate for a large chunk of the population. These results are interesting as they can be used to discard significant portions of the population in suicide attempt studies and focus on the groups that present much higher risk. Hence, our IBP with discrete observations is being able to obtain features that describe the hidden structure of the NESARC database and makes it possible to pin-point the people that have a higher risk of attempting suicide.

| # | Source Code | Description |
|---|---|---|
| 01 | S4AQ4A17 | Thought about committing suicide |
| 02 | S4AQ4A18 | Felt like wanted to die |
| 03 | S4AQ17A | Stayed overnight in hospital because of depression |
| 04 | S4AQ17B | Went to emergency room for help because of depression |
| 05 | S4AQ4A19 | Thought a lot about own death |
| 06 | S4AQ16 | Went to counselor/therapist/doctor/other person for help to improve mood |
| 07 | S4AQ18 | Doctor prescribed medicine/drug to improve mood/make you feel better |
| 08 | S4CQ15A | Stayed overnight in hospital because of dysthymia |
| 09 | S4AQ4A12 | Felt worthless most of the time for 2+ weeks |
| 10 | S4CQ15B | Went to emergency room for help because of dysthymia |
| 11 | S4AQ52 | Had arguments/friction with family, friends, people at work, or anyone else |
| 12 | S4AQ55 | Spent more time than usual alone because didn't want to be around people |
| 13 | S4AQ21C | Used medicine/drug on own to improve low mood prior to last 12 months |
| 14 | S4AQ21A | Ever used medicine/drug on own to improve low mood/make self feel better |
| 15 | S4AQ20A | Ever drank alcohol to improve low mood/make self feel better |
| 16 | S4AQ20C | Drank alcohol to improve mood prior to last 12 months |
| 17 | S4AQ56 | Couldn't do things usually did/wanted to do |
| 18 | S4AQ54 | Had trouble doing things supposed to do -like working, doing schoolwork, etc. |
| 19 | S4AQ11 | Any episode began after drinking heavily/more than usual |
| 20 | S4AQ15IR | Only/any episode prior to last 12 months began after drinking/drug use |

Table 1: Enumeration of the 20 selected questions in the experiments, sorted in decreasing order according to their mutual information with the 'attempted suicide' question.

## 5 Conclusions

In this paper, we have proposed a new model that combines the IBP with discrete observations using the multinomial-logit distribution. We have used the Laplace approximation to integrate out the weighting factors, which allows us to efficiently run the Gibbs sampler. We have applied our model to the NESARC database to find out the hidden features that characterize the suicide attempt risk. We

| Hidden features | | | | | | | Suicide attempt probability | | Number of cases | |
|---|---|---|---|---|---|---|---|---|---|---|
| | | | | | | | Train | Hold-out | Train | Hold-out |
| 1 | - | - | - | - | - | - | 6.74% | 5.55% | 430 | 8072 |
| - | 1 | - | - | - | - | - | 10.56% | 11.16% | 322 | 6083 |
| - | - | 1 | - | - | - | - | **3.72%** | **4.60%** | 457 | 8632 |
| - | - | - | 1 | - | - | - | **25.23%** | **22.25%** | 111 | 2355 |
| - | - | - | - | 1 | - | - | 8.64% | 9.69% | 301 | 5782 |
| - | - | - | - | - | 1 | - | 6.90% | 7.18% | 464 | 8928 |
| - | - | - | - | - | - | 1 | 14.29% | 14.18% | 91 | 1664 |
| - | - | 0 | 0 | - | - | - | 30.77% | 28.55% | 26 | 571 |
| - | - | 0 | 1 | - | - | - | **82.35%** | **61.95%** | 17 | 297 |
| - | - | 1 | 0 | - | - | - | **0.83%** | **0.87%** | 363 | 6574 |
| - | - | 1 | 1 | - | - | - | 14.89% | 16.52% | 94 | 2058 |
| - | - | 0 | 1 | - | - | 1 | 100.00% | 69.41% | 4 | 85 |
| 0 | - | 0 | 1 | - | - | - | 80.00% | 66.10% | 5 | 118 |
| 1 | - | 1 | 0 | - | 1 | 0 | 0.00% | 0.25% | 252 | 4739 |
| - | - | 1 | 0 | - | - | 0 | 0.33% | 0.63% | 299 | 5543 |
| 1 | - | 1 | 0 | - | - | - | **0.32%** | **0.41%** | 317 | **5807** |

Table 2: Probabilities of attempting suicide for different values of the latent feature vector, together with the number of subjects possessing those values. The symbol '-' denotes either 0 or 1. The 'train ensemble' columns contain the results for the 500 data points used to obtain the model, whereas the 'hold-out ensemble' columns contain the results for the remaining subjects.

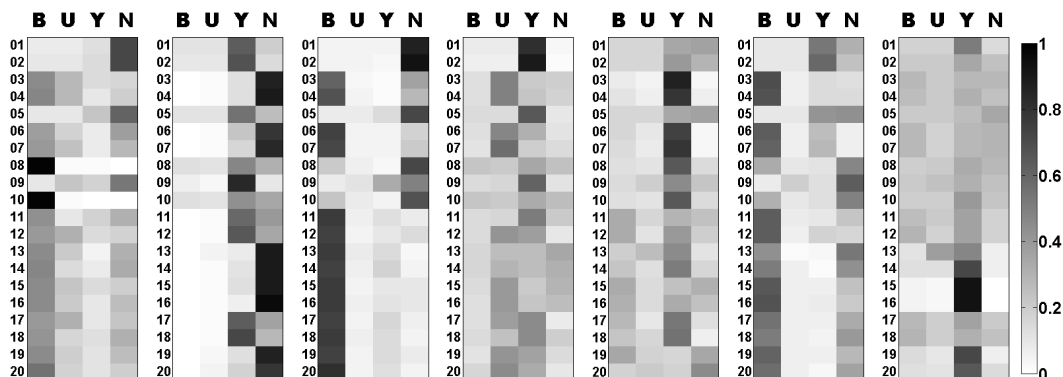

Figure 2: Probability of answering 'blank' (B), 'unknown' (U), 'yes' (Y) and 'no' (N) to each of the 20 selected questions, sorted as in Table 1, after 200 iterations of the Gibbs sampler. These probabilities have been obtained with the posterior mean weights $\mathbf{B}^d_{\mathrm{MAP}}$ , when only one of the seven latent features (sorted from left to right to match the order in Table 2) is active.

have analyzed how each of the seven inferred features contributes to the suicide attempt probability. We are developing a variational inference algorithm to be able to extend these remarkable results for larger fractions (subjects and questions) of the NESARC database.

**Acknowledgments**

Francisco J. R. Ruiz is supported by an FPU fellowship from the Spanish Ministry of Education, Isabel Valera is supported by the *Plan Regional-Programas I+D* of *Comunidad de Madrid* (AGES-CM S2010/BMD-2422), and Fernando Pérez-Cruz has been partially supported by a Salvador de Madariaga grant. The authors also acknowledge the support of *Ministerio de Ciencia e Innovación* of Spain (project DEIPRO TEC2009-14504-C02-00 and program Consolider-Ingenio 2010 CSD2008-00010 COMONSENS).

## Footnotes

[1]In a questionnaire of this size some questions are not asked when a previous question was answered in a predetermined way to reduce the burden of taking the survey. For example, if a person has never had a period of low mood, the attempt suicide question is not asked.

# References

[1] Summary of national strategy for suicide prevention: Goals and objectives for action, 2007. Available at: http://www.sprc.org/library/nssp.pdf.

[2] D. M. Blei and J. D. Lafferty. A correlated topic model of Science. *Annals of Applied Statistics*, 1(1):17–35, August 2007.

[3] S. Boyd and L. Vandenberghe. *Convex Optimization*. Cambridge University Press, March 2004.

[4] G. K. Brown, T. Ten Have, G. R. Henriques, S.X. Xie, J.E. Hollander, and A. T. Beck. Cognitive therapy for the prevention of suicide attempts: a randomized controlled trial. *Journal of the American Medical Association*, 294(5):563–570, 2005.

[5] T. L. Griffiths and Z. Ghahramani. The Indian Buffet Process: An introduction and review. *Journal of Machine Learning Research*, 12:1185–1224, 2011.

[6] D. A. Harville. *Matrix Algebra From a Statistician's Perspective*. Springer-Verlag, 1997.

[7] S. Haykin. *Adaptive Filter Theory*. Prentice Hall, 2002.

[8] R. C. Kessler, P. Berglund, G. Borges, M. Nock, and P. S. Wang. Trends in suicide ideation, plans, gestures, and attempts in the united states, 1990-1992 to 2001-2003. *Journal of the American Medical Association*, 293(20):2487–2495, 2005.

[9] K. Krysinska and G. Martin. The struggle to prevent and evaluate: application of population attributable risk and preventive fraction to suicide prevention research. *Suicide and Life-Threatening Behavior*, 39(5):548–557, 2009.

[10] D. J. C. MacKay. *Information Theory, Inference & Learning Algorithms*. Cambridge University Press, New York, NY, USA, 2002.

[11] J. J. Mann, A. Apter, J. Bertolote, A. Beautrais, D. Currier, A. Haas, U. Hegerl, J. Lonnqvist, K. Malone, A. Marusic, L. Mehlum, G. Patton, M. Phillips, W. Rutz, Z. Rihmer, A. Schmidtke, D. Shaffer, M. Silverman, Y. Takahashi, A. Varnik, D. Wasserman, P. Yip, and H. Hendin. Suicide prevention strategies: a systematic review. *The Journal of the American Medical Association*, 294(16):2064–2074, 2005.

[12] M. A. Oquendo, E. B. García, J. J. Mann, and J. Giner. Issues for DSM-V: suicidal behavior as a separate diagnosis on a separate axis. *The American Journal of Psychiatry*, 165(11):1383–1384, November 2008.

[13] K. Szanto, S. Kalmar, H. Hendin, Z. Rihmer, and J. J. Mann. A suicide prevention program in a region with a very high suicide rate. *Archives of General Psychiatry*, 64(8):914–920, 2007.

[14] M. Titsias. The infinite gamma-Poisson feature model. *Advances in Neural Information Processing Systems (NIPS)*, 19, 2007.

[15] J. Van Gael, Y. W. Teh, and Z. Ghahramani. The infinite factorial hidden Markov model. In *Advances in Neural Information Processing Systems (NIPS)*, volume 21, 2009.

[16] C. K. I. Williams and D. Barber. Bayesian classification with Gaussian Processes. *IEEE Transactions on Pattern Analysis and Machine Intelligence*, 20:1342–1351, 1998.

[17] S. Williamson, C. Wang, K. A. Heller, and D. M. Blei. The IBP Compound Dirichlet Process and its application to focused topic modeling. 11:1151–1158, 2010.

[18] M. A. Woodbury. The stability of out-input matrices. *Mathematical Reviews*, 1949.

